# Fusion with Diffusion for Robust Visual Tracking

**Yu Zhou**[1]*, **Xiang Bai**[1], **Wenyu Liu**[1], **Longin Jan Latecki**[2]

[1] Dept. of Electronics and Information Engineering, Huazhong Univ. of Science and Technology, P. R. China
[2] Dept. of Computer and Information Sciences, Temple Univ., Philadelphia, USA
{zhouyu.hust,xiang.bai}@gmail.com,liuwy@hust.edu.cn,latecki@temple.edu

## Abstract

A weighted graph is used as an underlying structure of many algorithms like semi-supervised learning and spectral clustering. If the edge weights are determined by a single similarity measure, then it hard if not impossible to capture all relevant aspects of similarity when using a single similarity measure. In particular, in the case of visual object matching it is beneficial to integrate different similarity measures that focus on different visual representations.

In this paper, a novel approach to integrate multiple similarity measures is proposed. First pairs of similarity measures are combined with a diffusion process on their tensor product graph (TPG). Hence the diffused similarity of each pair of objects becomes a function of joint diffusion of the two original similarities, which in turn depends on the neighborhood structure of the TPG. We call this process Fusion with Diffusion (FD). However, a higher order graph like the TPG usually means significant increase in time complexity. This is not the case in the proposed approach. A key feature of our approach is that the time complexity of the diffusion on the TPG is the same as the diffusion process on each of the original graphs. Moreover, it is not necessary to explicitly construct the TPG in our framework. Finally all diffused pairs of similarity measures are combined as a weighted sum. We demonstrate the advantages of the proposed approach on the task of visual tracking, where different aspects of the appearance similarity between the target object in frame $t-1$ and target object candidates in frame $t$ are integrated. The obtained method is tested on several challenge video sequences and the experimental results show that it outperforms state-of-the-art tracking methods.

## 1 Introduction

The considered problem has a simple formulation: Given are multiple similarities between the same set of $n$ data points, each similarity can be represented as a weighted graph. The goal is to combine them to a single similarity measure that best reflects the underlying data manifold. Since the set of nodes is the same, it is easy to combine the graphs into a single weighted multigraph, where there are multiple edges between the same pair of vertices representing different similarities. Then our task can be stated as finding a mapping from the multigraph to a weighted simple graph whose edge weights best represent the similarity of the data points. Of course, this formulation is not precise, since generally the data manifold is unknown, and hence it is hard to quantify the 'best'. However, it is possible to evaluate the quality of the combination experimentally in many applications, e.g., the tracking performance considered in this paper.

There are many possible solutions to the considered problem. One of the most obvious ones is a weighted linear combination of the similarities. However, this solution does not consider the similarity dependencies of different data points. The proposed approach aims to utilize the neighborhood structure of the multigraph in the mapping to the weighted simple graph.

Given two different similarity measures, we first construct their Tensor Product Graph (TPG). Then we jointly diffuse both similarities with a diffusion process on TPG. However, while the original graphs representing the two measures have $n$ nodes, their TPG has $n^2$ nodes, which significantly increases the time complexity of the diffusion on TPG. To address this problem, we introduce an iterative algorithm that operates on the original graphs and prove that it is equivalent to the diffusion on TPG. We call this process **Fusion with Diffusion (FD)**. FD is a generalization of the approached in [26], where only a single similarity measure is considered. While the diffusion process on TPG in [26] is used to enhances a single similarity measure, our approach aims at combining two different similarity measures so that they enhance and constrain each others.

Although algorithmically very different, our motivation is similar to co-training style algorithms in [5, 23, 24] where multiple cues are fused in an iterative learning process. The proposed approach is also related to the semi-supervised learning in [6, 7, 21, 28, 29]. For online tracking task, we only have the label information from the current frame, which can be regarded as the labeled data, and the label information in the next frame is unavailable, which can be regarded as unlabeled data. In this context, FD jointly propagates two similarities of the unlabeled data to the labeled data. The obtained new diffused similarity, can be then interpreted as the label probability over the unlabeled data. Hence from the point of view of visual tracking, but in the spirit of semi-supervised learning, our approach utilizes the unlabeled data from the next frame for improved visual similarity to the labeled data representing the tracked objets.

Visual tracking is an important issue in computer vision and has many practical applications. The challenges in designing a tracking system are often caused by shape deformation, occlusion, viewpoints variances, and background clutter. Different strategies have been proposed to obtain robust tracking systems. In [8, 12, 14, 16, 25, 27], matching based strategy is utilized. Discriminate appearance model of the target is extracted from the current frame, then the optimal target is estimated based on the distance/similarity between the appearance model and the candidate in the hypothesis set. Classification based strategies are introduced in [1, 2, 3, 4, 10, 11]. Tracking task is transformed into foreground and background binary classification problem in this framework. [15, 20] try to combine both of those two frameworks. In this paper, we focus on improving the distance/similarity measure to improve the matching based tracking strategy. Our motivation is similar to [12], where metric learning is proposed to improve the distance measure. However, different from [12], multiple cues are fused to improve the similarity in our approach. Moreover, the information from the forthcoming frame is also used to improve the similarity. This leads to more stable tracking performance than in [12].

Multiple cues fusion seem to be an effective way to improve the tracking performance. In [13], multiple feature fusion is implemented based on sampling the state space. In [20], the tracking task is formulated as the combination of different trackers, three different trackers are combined into a cascade. Different from those methods, we combine different similarities into a single similarity measure, which makes our method a more general for integrating various appearance models.

In summary, we propose a novel framework for integration of multiple similarity measures into a single consistent similarity measure, where the similarity of each pair of data points depends on their similarity to other data points. We demonstrate its superior performance on a challenging task of tracking by visual matching.

## 2   Problem Formulation

The problem of matching based visual tracking boils down to the following simple formulation. Given the target in frame $I_{t-1}$ which can be represented as image patch $\mathcal{I}_1$ enclosing the target, and the set of candidate target patches in frame $I_t$, $\mathcal{C} = \{\mathcal{I}_n | \ n = 2, ..., N\}$, the goal is to determine which patch in $\mathcal{C}$ corresponds to the target in frame $I_{t-1}$. Of course, one can make this setting more complicated, e.g., by considering more frames, but we consider this simple formulation in this paper.

The candidate set $\mathcal{C}$ is determined by the motion model, which is particularly simple in our setting. The size of all the image patches is fixed and the candidate set is composed of patches in frame $I_t$ inside a search radius $r$, i.e. $||c(\mathcal{I}_n) - c(\mathcal{I}_1)|| < r$, where $c$ is the 2-D coordinate of center position of the image patch.

Let $S$ be a similarity measure defined on the set of the image patches $V = \{\mathcal{I}_1\} \cup \mathcal{C}$, i.e., $S$ is a function from $V \times V$ into positive real numbers. Then our tracking goal can be formally stated as

$$\hat{\mathcal{I}} = \arg\max_{X \in \mathcal{C}} S(\mathcal{I}_1, X) \tag{1}$$

meaning that the patch in $\mathcal{C}$ with most similar appearance to patch $\mathcal{I}_1$ is selected as the target location in frame $t$.

Since the appearance of the target object changes, e.g., due to motion and lighting changes, single similarity measure is often not sufficient to identify the target in the next frame. Therefore, we consider a set of similarity measures $\mathcal{S} = \{S_1, \dots S_Q\}$, each $S_\alpha$ defined on $V \times V$ for $\alpha = 1, \dots, Q$. For example, in our experimental results, each image patch is represented with three histograms based on three different features, HOG[9], LBP[18], Haar-like feature[4], which lead to three different similarity measures. In other words, each pair of patches can be compared with respect to three different appearance features.

We can interpret each similarity measure $S_\alpha$ as the affinity matrix of a graph $G_\alpha$ whose vertex set is $V$, i.e., $S_\alpha$ a $N \times N$ matrix with positive entries, where $N$ is the cardinality of $V$. Then we can combine the graphs $G_\alpha$ into a single multigraph whose edge weights corresponds to different similarity measures $S_\alpha$.

However, in order to solve Eq. (1), we need a single similarity measure $S$. Hence we face a question how to combine the measures in $\mathcal{S}$ into a single similarity measure. We propose a two stage approach to answer this question. First, we combine pairs of similarity measures $S_\alpha$ and $S_\beta$ into a single measure $\mathrm{P}^*_{\alpha,\beta}$, which is a matrix of size $N \times N$. $\mathrm{P}^*_{\alpha,\beta}$ is defined in Section 3 and it is obtained with the proposed process called fusion with diffusion.

In the second stage we combine all $\mathrm{P}^*_{\alpha,\beta}$ for $\alpha, \beta = 1, \dots Q$ into a single similarity measure $S$ defined as a weighted matrix sum

$$S = \sum_{\alpha,\beta} \omega_\alpha \omega_\beta \mathrm{P}^*_{\alpha,\beta} \tag{2}$$

where $\omega_\alpha$ and $\omega_\beta$ are positive weights associated with measures $S_\alpha$ and $S_\beta$ defined in Section 5.

We also observe that in contrast to many tracking by matching methods, the combined measure $S$ is not only a function of similarities between $\mathcal{I}_1$ and the candidate patches in $\mathcal{C}$, but also of similarities of patches in $\mathcal{C}$ to each other.

## 3 Fusion with Diffusion

### 3.1 Single Graph on Consecutive Frames

Given a single graph $G_\alpha = (V, S_\alpha)$, a reversible Markov chain on $V$ can be constructed with the transition probability defined as

$$P_\alpha(i, j) = S_\alpha(i, j)/D_i \tag{3}$$

where $D_i = \sum_{j=1}^N S_\alpha(i, j)$ is the degree of each vertex. Then the transition probability $P_\alpha(i, j)$ inherits the positivity-preserving property $\sum_{j=1}^N P_\alpha(i, j) = 1$, $i = 1, ..., N$.

The graph $G_\alpha$ is fully connected graph in many applications. To reduce the influence of noisy points, i.e., cluttered background patches in tracking, a local transition probability is used:

$$(P_{k,\alpha})(i, j) = \begin{cases} P_\alpha(i, j) & j \in k\mathrm{NN}(i) \\ 0 & \text{otherwise} \end{cases} \tag{4}$$

Hence the number of non-zero elements in each row is not larger than $k$, which implies $\sum_{j=1}^n (P_{k,\alpha})(i, j) < 1$. This inequality is important in our framework, since it guarantees the converge of the diffusion process on the tensor product graph presented in the next section.

### 3.2 Tensor Product Graph of Two Similarities

Given are two graphs $G_\alpha = (V, P_{k,\alpha})$ and $G_\beta = (V, P_{k,\beta})$ defined in Sec. 3.1, we can define their **Tensor Product Graph (TPG)** as

$$G_\alpha \otimes G_\beta = (V \times V, \mathbb{P}), \tag{5}$$

where $\mathbb{P} = P_{k,\alpha} \otimes P_{k,\beta}$ is the Kronecker product of matrices defined as $\mathbb{P}(a,b,i,j) = P_{k,\alpha}(a,b) \, P_{k,\beta}(i,j)$. Thus, each entry of $\mathbb{P}$ relates four image patches. When $P_{k,\alpha}$ and $P_{k,\beta}$ are two $N \times N$ matrices, then $\mathbb{P}$ is a $N^2 \times N^2$ matrix. However, as we will see in the next subsection, we actually never compute $\mathbb{P}$ explicitly.

### 3.3 Diffusion Process on Tensor Product Graph

We utilize a diffusion process on TPG to combine the two similarity measures $P_{k,\alpha}$ and $P_{k,\beta}$. We begin with some notations. The $vec$ operator creates a column vector from a matrix $M$ by stacking the column vectors of $M$ below one another. More formally $vec : \mathbb{R}^{N \times N} \to \mathbb{R}^{N^2}$ is defined as $vec(M)_g = (M)_{ij}$, where $i = \lfloor (g-1)/N \rfloor + 1$ and $j = g \bmod N$. The inverse operator $vec^{-1}$ that maps a vector into a matrix is often called the reshape operator. We define a diagonal $N \times N$ matrix as

$$\Delta(i,i) = \left\{ \begin{array}{ll} 1 & i = 1 \\ 0 & \text{otherwise,} \end{array} \right. \tag{6}$$

Only the entry representing the patch $\mathcal{I}_1$ is set to one and all other entries are set to zero in $\Delta$.

We observe that $\mathbb{P}$ is the adjacency matrix of TPG $G_\alpha \otimes G_\beta$. We define a $q$-th iteration of the diffusion process on this graph as

$$\sum_{e=0}^{q} (\mathbb{P})^e vec(\Delta). \tag{7}$$

As shown in [26], this iterative process is guaranteed to converge to a nontrivial solution given by

$$\lim_{q \to \infty} \sum_{e=0}^{q} (\mathbb{P})^e vec(\Delta) = (\mathrm{I} - \mathbb{P})^{-1} vec(\Delta), \tag{8}$$

where I is a identity matrix. Following [26], we define

$$\mathrm{P}_{\alpha,\beta}^* = \mathrm{P}^* = vec^{-1}((\mathrm{I} - \mathbb{P})^{-1} vec(\Delta)) \tag{9}$$

We observe that our solution $\mathrm{P}^*$ is a $N \times N$ matrix.

We call the diffusion process to compute $\mathrm{P}^*$ a **Fusion with Diffusion (FD)** process, since diffusion on TPG $G_\alpha \otimes G_\beta$ is used to fuse two similarity measures $S_\alpha$ and $S_\beta$.

Since $\mathbb{P}$ is a $N^2 \times N^2$ matrix, FD process on TPG as defined in Eq. (7) may be computationally too demanding. To compute $\mathrm{P}^*$ effectively, instead of diffusing on TPG directly, we show in Section 3.4 that FD process on TPG is equivalent to an iterative process on $N \times N$ matrices only. Consequently, instead of an $O(n^6)$ time complexity, we obtain an $O(n^3)$ complexity. Then in Section 4 we further reduce it to an efficient algorithm with time complexity $O(n^2)$, which can be used in real time tracking algorithms.

### 3.4 Iterative Algorithm for Computing $\mathrm{P}^*$

We define $\mathrm{P}^1 = P_{(k,\alpha)} P_{(k,\beta)}^T$ and

$$\mathrm{P}^{q+1} = P_{k,\alpha} (P_{k,\alpha})^q (P_{k,\beta}^T)^q P_{k,\beta}^T + \Delta. \tag{10}$$

We iterate Eq.(10) until convergence, and as we prove in Proposition 1, we obtain

$$\mathrm{P}^* = \lim_{q \to \infty} \mathrm{P}^q = vec^{-1}((\mathrm{I} - \mathbb{P})^{-1} vec(\Delta)) \tag{11}$$

The iterative process in Eq.(10) is a generalization of the process introduced in [26]. Consequently, the following properties are simple extensions of the properties derived in [26]. However, we state them explicitly, since we combine two different affinity matrices, while [26] considers only a single matrix. In other words, we consider diffusion on TPG of two different graphs, while diffusion on TPG of a single graph with itself is considered in [26].

**Proposition 1**

$$vec\left(\lim_{q \to \infty} \mathrm{P}^{(q+1)}\right) = \lim_{q \to \infty} \sum_{e=0}^{q-1} \mathbb{P}^e vec(\Delta) = (\mathrm{I} - \mathbb{P})^{-1} vec(\Delta). \tag{12}$$

**Proof**: Eq.(10) can be rewritten as

$$
\begin{aligned}
\mathrm{P}^{(q+1)} &= P_{k,\alpha}\,(P_{k,\alpha})^q(P_{k,\beta}^T)^q\,P_{k,\beta}^T + \Delta \\
&= P_{k,\alpha}[P_{k,\alpha}\,(P_{k,\alpha})^{(q-1)}(P_{k,\beta}^T)^{(q-1)}\,P_{k,\beta}^T + \Delta]P_{k,\beta}^T + \Delta \\
&= (P_{k,\alpha})^2\,(P_{k,\alpha})^{(q-1)}(P_{k,\beta}^T)^{(q-1)}\,(P_{k,\beta}^T)^2 + P_{k,\alpha}\,\Delta\,P_{k,\beta} + \Delta \\
&= \cdots \\
&= (P_{k,\alpha})^q\,P_{k,\alpha}P_{k,\beta}^T\,(P_{k,\beta}^T)^q + (P_{k,\alpha})^{q-1}\,\Delta\,(P_{k,\beta}^T)^{q-1} + \cdots + \Delta \\
&= (P_{k,\alpha})^q\,P_{k,\alpha}P_{k,\beta}^T\,(P_{k,\beta}^T)^q + \sum_{e=0}^{q-1}(P_{k,\alpha})^e\,\Delta\,(P_{k,\beta}^T)^e
\end{aligned}
\tag{13}
$$

**Lemma 1** $\lim_{q\to\infty}(P_{k,\alpha})^q\,P_{k,\alpha}P_{k,\beta}^T\,(P_{k,\beta}^T)^q = 0$

**Proof**: It suffices to show that $(P_{k,\alpha})^q$ and $(P_{k,\beta}^T)^q$ go to 0, when $q \to \infty$. This is true if and only if every eigenvalue of $P_{k,\alpha}$ and $P_{k,\beta}$ is less than one in absolute value. Since $P_{k,\alpha}$ and $P_{k,\beta}$ has nonnegative entries, this holds if its row sums are all less than one. As described in Sec.3.1, we have $\sum_{b=1}^{N}(P_{k,\alpha})_{a,b} < 1$ and $\sum_{j=1}^{N}(P_{k,\beta})_{i,j} < 1$.

Lemma 1 shows that the first summand in Eq.(13) converges to zero, and consequently we have

$$
\lim_{q\to\infty}\mathrm{P}^{(q+1)} = \lim_{q\to\infty}\sum_{e=0}^{q-1}(P_{k,\alpha})^e\,\Delta\,(P_{k,\beta}^T)^e.
\tag{14}
$$

**Lemma 2** $vec\left((P_{k,\alpha})^e\,\Delta\,(P_{k,\beta}^T)^e\right) = (\mathrm{P})^e vec(\Delta)$ for $e = 1, 2, \ldots$.

**Proof**: Our proof is by induction. Suppose $(\mathrm{P})^l vec(\Delta)=vec\left((P_{k,\alpha})^l\,\Delta\,(P_{k,\beta}^T)^l\right)$ is true for $e = l$, then for $e = l + 1$ we have

$$
\begin{aligned}
(\mathrm{P})^{l+1}vec(\Delta) &= \mathrm{P}\,\mathrm{P}^l vec(\Delta) = vec\left(P_{k,\alpha}\,vec^{-1}(\mathrm{P}^l vec(\Delta))\,P_{k,\beta}^T\right) \\
&= vec\left(P_{k,\alpha}\,((P_{k,\alpha})^l\,\Delta\,(P_{k,\beta}^T)^l)\,P_{k,\beta}^T\right) \\
&= vec\left((P_{k,\alpha})^{l+1}\,\Delta\,(P_{k,\beta}^T)^{l+1}\right)
\end{aligned}
$$

and the proof of Lemma 2 is complete.

By Lemma 1 and Lemma 2, we obtain that

$$
vec\left(\sum_{e=0}^{q-1}(P_{k,\alpha})^e\,\Delta\,(P_{k,\beta}^T)^e\right) = \sum_{e=0}^{q-1}(\mathrm{P})^e vec(\Delta).
\tag{15}
$$

The following useful identity holds for the Kronecker Product [22]:

$$
vec(P_{k,\beta}\Delta P_{k,\alpha}^T) = (P_{k,\alpha} \otimes P_{k,\beta})vec(\Delta) = (\mathbb{P})vec(\Delta)
\tag{16}
$$

Putting together (14), (15), (16), we obtain

$$
vec\left(\lim_{q\to\infty}\mathrm{P}^{(q+1)}\right) = vec\left(\lim_{q\to\infty}\sum_{e=0}^{q-1}(P_{k,\alpha})^e\,\Delta\,(P_{k,\beta}^T)^e\right)
\tag{17}
$$

$$
= \lim_{q\to\infty}\sum_{e=0}^{q-1}\mathbb{P}^e vec(\Delta) = (\mathrm{I} - \mathbb{P})^{-1}vec(\Delta)=vec(\mathrm{P}^*).
\tag{18}
$$

This proves Proposition 1.

We now show how FD could improve the original similarity measures. Suppose we have two similarity measures $S_\alpha$ and $S_\beta$. $\mathcal{I}_1$ denotes the image patch enclosing the target in frame $t-1$. According to $S_\alpha$, there are many patches in frame $t$ that have nearly equal similarity to $\mathcal{I}_1$ with patch $\mathcal{I}_n$ being most similar to $\mathcal{I}_1$, while according to $S_\beta$, $\mathcal{I}_1$ is clearly more similar to $\mathcal{I}_m$ in frame $t$. Then the proposed diffusion will enhance the similarity $S_\beta(\mathcal{I}_1, \mathcal{I}_m)$, since it will propagate faster the $S_\beta$ similarity of $\mathcal{I}_1$ to $\mathcal{I}_m$ than to the other patches. In contrast, the $S_\alpha$ similarities will propagate with similar speed. Consequently, the final joint similarity $\mathrm{P}^*$ will have $\mathcal{I}_m$ as the most similar to $\mathcal{I}_1$.

---

**Algorithm 1:** Iterative Fusion with Diffusion Process

---

**Input**: Two matrices $P_{k,\alpha}, P_{k,\beta} \in \mathbb{R}^{N \times N}$
**Output**: Diffusion result $P^* \in \mathbb{R}^{N \times N}$

**1** Compute $P^* = \Delta$.
**2** Compute $\mathbf{u}_\alpha$ = first column of $P_{k,\alpha}$, $\mathbf{u}_\beta$ = first column of $P_{k,\beta}$
**3** Compute $P^* \leftarrow P^* + \mathbf{u}_\alpha \mathbf{u}_\beta^T$.
**4 for** $i = 2, 3, \ldots$ **do**
**5** $\quad$ Compute $\mathbf{u}_\alpha \leftarrow P_{k,\alpha} \mathbf{u}_\alpha$
**6** $\quad$ Compute $\mathbf{u}_\beta \leftarrow P_{k,\beta} \mathbf{u}_\beta$
**7** $\quad$ Compute $P^* \leftarrow P^* + \mathbf{u}_\alpha \mathbf{u}_\beta^T$
**8 end**

---

## 4 FD Algorithm

To effectively compute $P^*$, we propose an iterative algorithm that takes the advantage of the structure of matrix $\Delta$. Let $\mathbf{u}_\alpha$ be a $N \times 1$ vector containing the first column of $P_{k,\alpha}$. We write $P_{k,\alpha} = [\mathbf{u}_\alpha | R]$ and $P_{k,\alpha} \Delta = [\mathbf{u}_\alpha | 0]$. It follows then that $P_{k,\alpha} \Delta P_{k,\beta}^T = \mathbf{u}_\alpha \mathbf{u}_\beta^T$. Furthermore, if we denote $(P_{k,\alpha})^j \Delta (P_{k,\beta}^T)^j = \mathbf{u}_{\alpha,j} \mathbf{u}_{\beta,j}^T$, with $\mathbf{u}_{\alpha,j}$ being $N \times 1$, and $\mathbf{u}_{\beta,j}^T$ being $1 \times N$, it follows that

$$P_{k,\alpha}^{j+1} \Delta (P_{k,\beta}^T)^{j+1} = P_{k,\alpha}(P_{k,\alpha}^j \Delta (P_{k,\beta}^T)^j)P_{k,\beta}^T = P_{k,\alpha}\mathbf{u}_{\alpha,j}\mathbf{u}_{\beta,j}^T P_{k,\beta}^T$$
$$= (P_{k,\alpha}\mathbf{u}_{\alpha,j})(P_{k,\beta}\mathbf{u}_{\beta,j})^T = \mathbf{u}_{\alpha,j+1}\mathbf{u}_{\beta,j+1}^T.$$

Hence, we replaced one of the two $N \times N$ matrix products with one matrix product between an $N \times N$ matrix and $N \times 1$ vector, and the other with a product of an $N \times 1$ by an $1 \times N$ vector. This reduces the complexity of our algorithm from $O(n^3)$ to $O(n^2)$.

The final algorithm is shown in Alg. 1.

## 5 Weight Estimation

The weight $\omega_\alpha$ of measure $S_\alpha$ is proportional to how well $S_\alpha$ is able to distinguish the target $\mathcal{I}_1$ in frame $I_{t-1}$ from the background surrounding the target. Let $\{\mathcal{B}_h | \ h = 1, ..., H\}$ be a set of background patches surrounding the target $\mathcal{I}_1$ in frame $I_{t-1}$. The weight of $S_\alpha$ is defined as

$$\omega_\alpha = \frac{1}{\frac{1}{H}\sum_{h=1}^{H} S_\alpha(\mathcal{I}_1, \mathcal{B}_h)} \tag{19}$$

Thus, the smaller the values of $S_\alpha$, the larger is the weight $\omega_\alpha$. The weights of all similarity measures are normalized so that $\sum_{\alpha=1}^{Q} \omega_\alpha = 1$. The weights are computed for every frame in order to accommodate appearance changes of the tracked object.

## 6 Experimental Results

We validate our tracking algorithm on eight challenging videos from [4] and [17]: *Sylvester*, *Coke Can*, *Tiger1*, *Cliff Bar*, *Coupon Book*, *Surfer*, and *Tiger2*, *PETS01D1*. We compare our method with six famous state-of-the-art tracking algorithms including Multiple Instance Learning tracker (MIL) [4], Fragment tracker(Frag) [1], IVT [19], Online Adaboost tracker (OAB) [10], SemiBoost tracker (Semi) [11], Mean-Shift (MS) tracker, and a simple weighted linear sum of multiple cues (Linear). For the comparison methods, we run source code of Semi, Frag, MIL, IVT and OAB supplied by the authors on the testing videos and use the parameters mentioned in their papers directly. For MS, we implement it based on OpenCV. For Linear, we use three kinds of image features to get the affinity and then simply calculate the average affinity and use the diffusion process mentioned in [26]. *Note that all the parameters in our algorithm were fixed for all the experiments*.

In our experiments, HOG[9], LBP[18] and Haar-like[4] features are used to represent the image patches. Hence each pair of patches is compared with three different similarities based on histograms

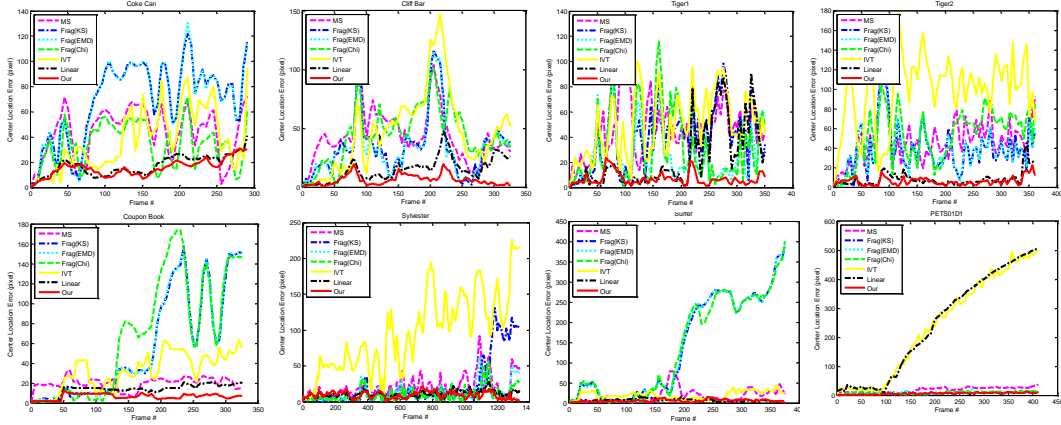

Figure 1: *Center Location Error* (CLE) versus frame number

of HOG, LBP, and Haar-like feature. For the experimental parameters, we set $r = 15$ pixels, $H = 300$, $k = 12$ and the iteration number in Alg. 1 is set to 200.

To impartially and comprehensively compare our algorithm with other state-of-the-art trackers, we used two kinds of quantitative comparisons *Average Center Location Error* (ACLE) and Precision Score [4]. The results are shown in Table 1 and Table 2, respectively. Two kinds of curve evaluation methodologies are also used *Center Location Error* (CLE) curve and Precision Plots curve[1]. The results are shown in Fig.1 and Fig.2, respectively.

Table 1: *Average Center Location Error* (ACLE measured in pixels). Red color indicates best performance, Blue color indicates second best, Green color indicates the third best

| Video | MS | OAB | IVT | Semi | Frag1 | Frag2 | Frag3 | MIL | Linear | **our** |
|---|---|---|---|---|---|---|---|---|---|---|
| *Coke Can* | 43.7 | 25.0 | 37.3 | 40.5 | 69.1 | 69.0 | 34.1 | 31.9 | 16.8 | 15.4 |
| *Cliff Bar* | 43.8 | 34.6 | 47.1 | 57.2 | 34.7 | 34.0 | 44.8 | 14.2 | 15.0 | 6.1 |
| *Tiger 1* | 45.5 | 39.8 | 50.2 | 20.9 | 39.7 | 26.7 | 31.1 | 7.6 | 23.8 | 6.9 |
| *Tiger2* | 47.6 | 13.2 | 98.5 | 39.3 | 38.6 | 38.8 | 51.9 | 20.6 | 6.5 | 5.7 |
| *Coup. Book* | 20.0 | 17.7 | 32.2 | 65.1 | 55.9 | 56.1 | 67.0 | 19.8 | 13.6 | 6.5 |
| *Sylvester* | 20.0 | 35.0 | 96.1 | 21.0 | 23.0 | 12.2 | 10.1 | 11.4 | 10.5 | 9.3 |
| *Surfer* | 17.0 | 13.4 | 19.0 | 9.3 | 140.1 | 139.8 | 138.6 | 7.7 | 6.5 | 5.5 |
| *PETS01D1* | 18.1 | 7.1 | 241.8 | 158.9 | 6.7 | 7.2 | 9.5 | 11.7 | 245.4 | 6.0 |

Table 2: *Precision Score* (precision at the fixed threshold of 15). Red color indicates best performance, Blue color indicates second best, Green color indicates the third best.

| Video | MS | OAB | IVT | Semi | Frag1 | Frag2 | Frag3 | MIL | Linear | **our** |
|---|---|---|---|---|---|---|---|---|---|---|
| *Coke Can* | 0.11 | 0.21 | 0.15 | 0.18 | 0.09 | 0.09 | 0.17 | 0.24 | 0.36 | 0.46 |
| *Cliff Bar* | 0.08 | 0.21 | 0.19 | 0.34 | 0.20 | 0.23 | 0.12 | 0.79 | 0.52 | 0.95 |
| *Tiger 1* | 0.05 | 0.17 | 0.03 | 0.52 | 0.21 | 0.38 | 0.38 | 0.90 | 0.54 | 0.91 |
| *Tiger 2* | 0.06 | 0.65 | 0.01 | 0.44 | 0.09 | 0.09 | 0.12 | 0.66 | 0.89 | 0.95 |
| *Coupon Book* | 0.16 | 0.18 | 0.21 | 0.41 | 0.39 | 0.39 | 0.39 | 0.23 | 0.53 | 1.00 |
| *Sylvester* | 0.46 | 0.30 | 0.06 | 0.53 | 0.72 | 0.78 | 0.81 | 0.76 | 0.86 | 0.90 |
| *Surfer* | 0.59 | 0.61 | 0.40 | 0.89 | 0.19 | 0.21 | 0.23 | 0.93 | 1.00 | 1.00 |
| *PETS01D1* | 0.38 | 1.00 | 0.01 | 0.29 | 0.99 | 0.97 | 0.95 | 0.80 | 0.02 | 1.00 |

**Comparison to matching based methods**: MS, IVT, Frag and Linear are all matching based tracking algorithms. In MS, famous Bhattacharyya coefficient is used to measure the distance between histogram distributions; for Frag, we test it under three different measurement strategies: the

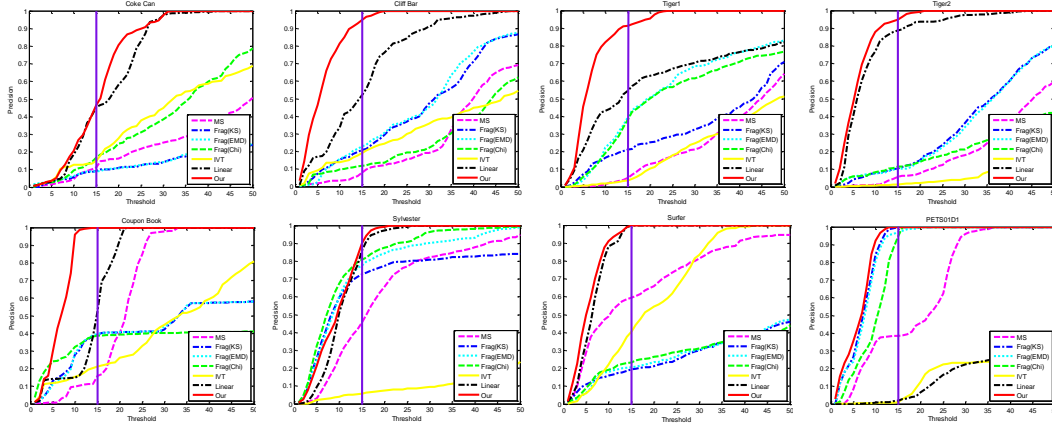

Figure 2: *Precision Plots*. The threshold is set to 15 in our experiments.

Kolmogorov-Smirnov statistic, EMD, and Chi-Square distance, represented as Frag1, Frag2, Frag3 in Table 1 and Table 2, respectively. For Linear Combination, the average similarity is used and the diffusion process in [26] is used to improve the similarity measure. Our FD approach clearly outperforms the other approaches, as shown in Table1 and Table2. Our tracking results achieve the best performance in all the testing videos, especially for the Precision Plots shown in Table 2. Even though we set the threshold to 15, which is more challenging for all the trackers, we still get three 1.00 scores. In some videos like *sylvester* and *PETS01D1*, Frag achieves comparable results with our method, but it works badly in other videos which means that specific distance measure can only work on some special cases but our fusion framework is robust for all the challenges that appear in the videos. Our method is always batter than Linear Combination, which means that the fusion with diffusion can really improve the tracking performance. The stability of our method can be clearly seen in the plots of location error as the function of frame number in Fig.1. Our tracking results are always stable, which means that we do not lose the target in the whole tracking process. This is also reflected in the fact that our Precision is always batter than all the other methods under different thresholds as shown in Fig.2.

**Comparison to classification based methods**: MIL and OAB are both classification based tracking algorithms. For OAB, on-line Adaboost is used to train the classifier for the foreground and background classification. MIL combines multiple instance learning with on-line Adaboost. Haar-like features are used in both methods. Again our method outperforms those two methods as can be seen in Table1 and Table 2.

**Comparison to semi-supervised learning based methods**: SemiBoost combines semi-supervised learning with on-line Adaboost. Our method is also similar to semi-supervised learning for we build the graph model on consecutive frames, which means that both of our method and SemiBoost use the information from the forthcoming frame. Our method is always better than SemiBoost as shown in Table 1 and Table 2.

## 7 Conclusions

In this paper, a novel Fusion with Diffusion process is proposed for robust visual tracking. Pairs of similarity measures are fused into a single similarity measure with a diffusion process on the tensor product of two graphs determined by the two similarity measures. The proposed method has time complexity of $O(n^2)$, which makes it suitable for real time tracking. It is evaluated on several challenging videos, and it significantly outperforms a large number of state-of-the-art tracking algorithms.

### Acknowledgments

We would like to thank all the authors for releasing their source codes and testing videos, since they made our experimental evaluation possible. This work was supported by NSF Grants IIS-0812118, BCS-0924164, OIA-1027897, and by the National Natural Science Foundation of China (NSFC) Grants 60903096, 61222308 and 61173120.

## Footnotes

*Part of this work was done while the author was visiting Temple University

[1]More details about the meaning of Precision Plots can be found in [4]

# References

[1] A. Adam, E. Rivlin, and I. Shimshoni. Robust fragment-based tracking using the integral histogram. In *IEEE Computer Society Conference on Computer Vision and Pattern Recognition(CVPR)*, pages 798–805, 2006.

[2] S. Avidan. Support vector tracking. *IEEE Transactions on Pattern Analysis and Machine Intelligence*, 26(8):1064–1072, 2004.

[3] S. Avidan. Ensemble tracking. *IEEE Transactions on Pattern Analysis and Machine Intelligence*, 29(2):261–271, 2007.

[4] B. Babenko, M. Yang, and S. Belongie. Robust object tracking with online multiple instance learning. *IEEE Transactions on Pattern Analysis and Machine Intelligence*, 33(8):1619–1632, 2011.

[5] X. Bai, B. Wang, C. Yao, W. Liu, and Z. Tu. Co-transduction for shape retrieval. *IEEE Transactions on Image Processing*, 21(5):2747–2757, 2012.

[6] X. Bai, X. Yang, L. J. Latecki, W. Liu, and Z. Tu. Learning context sensitive shape similarity by graph transduction. *IEEE Transactions on Pattern Analysis and Machine Intelligence*, 32(5):861–874, 2010.

[7] M. Belkin and P. Niyogi. Semi-supervised learning on riemannian manifolds. *Machine Learning*, 56(special Issue on clustering):209–239, 2004.

[8] D. Comaniciu, V. R. Member, and P. Meer. Kernel-based object tracking. *IEEE Transactions on Pattern Analysis and Machine Intelligence*, 25(5):564–575, 2003.

[9] N. Dalal and B. Triggs. Histograms of oriented gradients for human detection. In *IEEE Computer Society Conference on Computer Vision and Pattern Recognition(CVPR)*, pages 886–893, 2005.

[10] H. Grabner, M. Grabner, and H. Bischof. Real-time tracking via on-line boosting. In *British Machine Vision Conference(BMVC)*, pages 47–56, 2006.

[11] H. Grabner, C. Leistner, and H. Bischof. Semi-supervised on-line boosting for robust tracking. In *European Conference on Computer Vision(ECCV)*, pages 234–247, 2008.

[12] N. Jiang, W. Liu, and Y. Wu. Learning adaptive metric for robust visual tracking. *IEEE Transactions on Image Processing*, 20(8):2288–2300, 2011.

[13] J. Kwon and K. M. Lee. Visual tracking decomposition. In *IEEE Computer Society Conference on Computer Vision and Pattern Recognition(CVPR)*, 2010.

[14] J. Lim, D. Ross, R.-S. Lin, and M.-H. Yang. Incremental learning for visual tracking. In *Advances in Neural Information Processing Systems (NIPS)*, 2005.

[15] R. Liu, J. Cheng, and H. Lu. A robust boosting tracker with minimum error bound in a co-training framework. In *IEEE Interestial Conference on Computer Vision(ICCV)*, 2009.

[16] X. Mei and H. Ling. Robust visual tracking and vehicle classification via sparse representation. *IEEE Transactions on Pattern Analysis and Machine Intelligence*, 33(11):2259–2272, 2011.

[17] X. Mei, H. Ling, Y. Wu, E. Blasch, and L. Bai. Minimum error bounded efficient l1 tracker with occlusion detection. In *IEEE Conf. on Computer Vision and Pattern Recognition (CVPR)*, 2011.

[18] T. Ojala, M. Pietikäinen, and T. Mäenpää. Multiresolution gray-scale and rotation invariant texture classification with local binary patterns. *IEEE Transactions on Pattern Analysis and Machine Intelligence*, 24(7):971–987, 2002.

[19] D. Ross, J. Kim, R.-S. Lin, and M.-H. Yang. Incremental learning for robust visual tracking. *International Journal of Computer Vision*, 77(1):125–141, 2008.

[20] J. Santner, C. Leistner, A. Saffari, T. Pock, and H. Bischof. Prost: Parallel robust online simple tracking. In *IEEE Computer Society Conference on Computer Vision and Pattern Recognition(CVPR)*, 2010.

[21] K. Sinha and M.Belkin. Semi-supervised learning using sparse eigenfunction bases. In *Advances in Neural Information Processing Systems(NIPS)*, 2009.

[22] S. Vishwanathan, N. Schraudolph, R. Kondor, and K. Borgwardt. Graph kernels. *Journal of Machine Learning Research*, 11(4):1201–1242, 2010.

[23] B. Wang, J. Jiang, W. Wang, Z.-H. Zhou, and Z. Tu. Unsupervised metric fusion by cross diffusion. In *IEEE Computer Society Conference on Computer Vision and Pattern Recognition(CVPR)*, 2012.

[24] W. Wang and Z. Zhou. A new analysis of co-training. In *Internal Conference on Machine Learning(ICML)*, 2010.

[25] Y. Wu and J. Fan. Contextual flow. In *IEEE Computer Society Conference on Computer Vision and Pattern Recognition(CVPR)*, 2009.

[26] X. Yang and L. J. Latecki. Affinity learning on a tensor product graph with applications to shape and image retrieval. In *IEEE Computer Society Conference on Computer Vision and Pattern Recognition(CVPR)*, 2011.

[27] W. Zhong, H. Lu, and M.-H. Yang. Robust object tracking via sparsity-based collaborative model. In *Proceedings of IEEE Conference on Computer Vision and Pattern Recognition (CVPR)*, 2012.

[28] D. Zhou, O. Bousquet, T. Lal, J. Weston, and B. Scholkopf. Learning with local and global consistency. In *Advances in Neural Information Processing Systems (NIPS)*, 2004.

[29] X. Zhu. Semi-supervised learning literature survey. In *Technical Report 1530, Department of Computer Sciences, University of Wisconsin, Madison*, 2005.

